# Symplectic Nonlinear Component Analysis

**Lucas C. Parra**
Siemens Corporate Research
755 College Road East, Princeton, NJ 08540
lucas@scr.siemens.com

## Abstract

Statistically independent features can be extracted by finding a factorial representation of a signal distribution. Principal Component Analysis (PCA) accomplishes this for linear correlated and Gaussian distributed signals. Independent Component Analysis (ICA), formalized by Comon (1994), extracts features in the case of linear statistical dependent but not necessarily Gaussian distributed signals. Nonlinear Component Analysis finally should find a factorial representation for nonlinear statistical dependent distributed signals. This paper proposes for this task a novel feed-forward, information conserving, nonlinear map - the explicit symplectic transformations. It also solves the problem of non-Gaussian output distributions by considering single coordinate higher order statistics.

## 1 Introduction

In previous papers Deco and Brauer (1994) and Parra, Deco, and Miesbach (1995) suggest volume conserving transformations and factorization as the key elements for a nonlinear version of Independent Component Analysis. As a general class of volume conserving transformations Parra et al. (1995) propose the symplectic transformation. It was defined by an implicit nonlinear equation, which leads to a complex relaxation procedure for the function recall. In this paper an explicit form of the symplectic map is proposed, overcoming thus the computational problems.

In order to correctly measure the factorization criterion for non-Gaussian output distributions, higher order statistics has to be considered. Comon (1994) includes in the linear case higher order cumulants of the output distribution. Deco and Brauer (1994) consider multi-variate, higher order moments and use them in the case of nonlinear volume conserving transformations. But the calculation of multi-coordinate higher moments is computational expensive.

The factorization criterion for statistical independence can be expressed in terms of minimal mutual information. Considering only volume conserving transformations allows to concentrate on single coordinate statistics, which leads to an important reduction of computational complexity. So far, this approach (Deco & Schürman, 1994; Parra et al., 1995) has been restricted to second order statistic. The present paper discusses the use of higher order cumulants for the estimation of the single coordinate output distributions. The single coordinate entropies measured by the proposed technique match the entropies of the sampled data more accurately. This leads in turns to better factorization results.

## 2    Statistical Independence

More general than decorrelation used in PCA the goal is to extract statistical independent features from a signal distribution $p(\mathbf{x})$. We look for a deterministic transformation on $\Re^n$: $\mathbf{y} = f(\mathbf{x})$ which generates a factorial representation $p(\mathbf{y}) = \prod_i p(y_i)$, or at least a representation where the individual coordinates $p(y_i)$ of the output variable $\mathbf{y}$ are "as factorial as possible". This can be accomplished by minimizing the mutual information $MI[p(\mathbf{y})]$.

$$0 \leq MI[p(\mathbf{y})] = \sum_{i=1}^{n} H[p(y_i)] - H[p(\mathbf{y})], \tag{1}$$

since $MI[p(\mathbf{y})] = 0$ holds if $p(\mathbf{y})$ is factorial. The mutual information can be used as a measure of "independence". The entropies $H$ in the definition (1) are defined as usual by $H[p(y)] = -\int_{-\infty}^{\infty} p(y) \ln p(y) \, dy$.

As in linear PCA we select volume conserving transformations, but now without restricting ourselves to linearity. In the noise-free case of reversible transformations volume conservation implies conservation of entropy from the input $\mathbf{x}$ to the output $\mathbf{y}$, *i.e.* $H[p(\mathbf{y})] = H[p(\mathbf{x})] = const$ (see Papoulis, 1991). The minimization of mutual information (1) reduces then to the minimization of the single coordinate output entropies $H[p(y_i)]$. This substantially simplifies the complexity of the problem, since no multi-coordinate statistics is required.

### 2.1    Measuring the Entropy with Cumulants

With an upper bound minimization criterion the task of measuring entropies can be avoided (Parra et al., 1995):

$$H[p(y_i)] \leq \frac{1}{2} \ln(2\pi e) + \frac{1}{2} \ln \sigma_i^2. \tag{2}$$

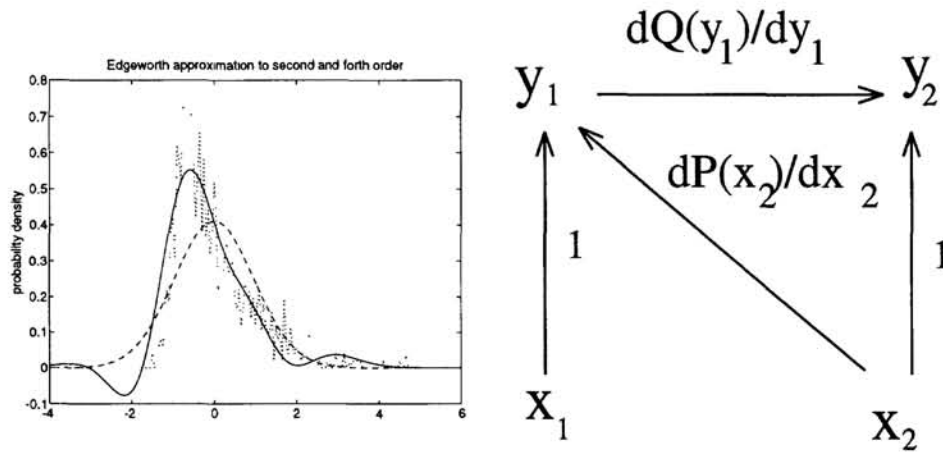

Figure 1: LEFT: Doted line: exponential distribution with additive Gaussian noise sampled with 1000 data points. (noise-variance/decay-constant = 0.2). Dashed line: Gaussian approximation equivalent to the Edgeworth approximation to second order. Solid line: Edgeworth approximation including terms up to fourth order. RIGHT: Structure of the volume conserving explicit symplectic map.

The minimization of the individual output coordinate entropies $H[p(y_i)]$ simplifies to the minimization of output variances $\sigma_i$. For the validity of that approach it is crucial that the map $\mathbf{y} = f(\mathbf{x})$ transforms the arbitrary input distribution $p(\mathbf{x})$ into a Gaussian output distribution. But volume conserving and continuous maps can not transform arbitrary distributions into Gaussians. To overcome this problem one includes statistics - higher than second order - to the optimization criterion.

Comon (1994) suggests to use the Edgeworth expansion of a probability distribution. This leads to an analytic expression of the entropy in terms of measurable higher order cumulants. Edgeworth expands the multiplicative correction to the best Gaussian approximation of the distribution in the orthonormal basis of Hermite polynomials $h_\alpha(y)$. The expansion coefficients are basically given by the cumulants $c_\alpha$ of distribution $p(y)$. The Edgeworth expansions reads for a zero-mean distribution with variance $\sigma^2$, (see Kendall & Stuart, 1969)

$$
\begin{aligned}
p(y) &= \frac{1}{\sqrt{2\pi}\sigma} e^{-\frac{y^2}{2\sigma^2}} f(y) \\
f(y) &= 1 + \frac{c_3}{6\sigma^3} h_3(\tfrac{y}{\sigma}) + \frac{c_4}{24\sigma^4} h_4(\tfrac{y}{\sigma}) + \frac{c_5}{120\sigma^5} h_5(\tfrac{y}{\sigma}) + \dots
\end{aligned}
\tag{3}
$$

Note, that by truncating this expansion at a certain order, we obtain an approximation $p_{app}(y)$, which is not strictly positive. Figure 1, left shows a sampled exponential distribution with additive Gaussian noise.

By cutting expansion (3) at fourth order, and further expanding the logarithm in definition of entropy up to sixth order, Comon (1994) approximates the entropy by,

$$H[p(y)_{app}] \approx \frac{1}{2}\ln(2\pi e) + \ln\sigma - \frac{1}{12}\frac{c_3^2}{\sigma^6} - \frac{1}{48}\frac{c_4^2}{\sigma^8} - \frac{7}{48}\frac{c_3^4}{\sigma^{12}} + \frac{1}{8}\frac{c_3^2}{\sigma^6}\frac{c_4}{\sigma^4} \qquad (4)$$

We suggest to use this expression to minimize the single coordinate entropies in the definition of the mutual information (1).

## 2.2   Measuring the Entropy by Estimating an Approximation

Note that (4) could only be obtained by truncating the expansion (3). It is therefore limited to fourth order statistic, which might be not enough for a satisfactory approximation. Besides, the additional approximation of the logarithm is accurate only for small corrections to the best Gaussian approximation, *i.e.* for $f(y) \approx 1$. For distributions with non-Gaussian tails the correction terms might be rather large and even negative as noted above. We therefore suggest alternatively, to measure the entropy by estimating the logarithm of the approximated distribution $\ln p_{app}(y)$ with the given data points $y_\nu$ and using Edgeworth approximation (3) for $p_{app}(y)$,

$$H[p(y)] \approx -\frac{1}{N}\sum_{\nu=1}^{N}\ln p_{app}(y_\nu) = const + \ln\sigma - \frac{1}{N}\sum_{\nu=1}^{N}\ln f(y_\nu) \qquad (5)$$

Furthermore, we suggest to correct the truncated expansion $p_{app}$ by setting $f_{app}(y) \to 0$ for all $f_{app}(y) < 0$. For the entropy measurement (5) there is in principle no limitation to any specific order.

In table 1 the different measures of entropy are compared. The values in the row labeled 'partition' are measured by counting the numbers $n(i)$ of data points falling in equidistant intervals $i$ of width $\Delta y$ and summing $-p(i)\Delta y \ln p(i)$ over all intervals, with $p(i)\Delta y = n(i)/N$. This gives good results compared to the theoretical values only because of the relatively large sampling size. These values are presented here in order to have an reliable estimate for the case of the exponential distribution, where cumulant methods tend to fail.

The results for the exponential distribution show the difficulty of the measurement proposed by Comon, whereas the estimation measurement given by equation (5) is stable even when considering (for this case) unreliable 5th and 6th order cumulants. The results for the symmetric-triangular and uniform distribution demonstrate the insensibility of the Gaussian upper bound for the example of figure 2. A uniform squared distribution is rotated by an angle $\alpha$. On the abscissa and ordinate a triangular or uniform distribution are observed for the different angles $\alpha = \Pi/4$ or $\alpha = 0$ respectively. The approximation of the single coordinate entropies with a Gaussian measure is in both cases the same. Whereas measurements including higher order statistics correctly detect minimal entropy (by fixed total information) for the uniform distribution at $\alpha = 0$.

## 3   Explicit Symplectic Transformation

Different ways of realizing a volume conserving transformation that guarantees $H[p(\mathbf{x})] = H[p(\mathbf{x})]$ have been proposed (Deco & Schürman, 1994; Parra et al.,

| Measured entropy of sampled distributions | Gauss | uniform | triangular symmetric | exponential + Gauss noise |
|---|---|---|---|---|
| partition | $1.35 \pm .02$ | $.024 \pm .006$ | $.14 \pm .02$ | $1.31 \pm .03$ |
| Gaussian upper bound (2) | $1.415 \pm .02$ | $.18 \pm .016$ | $.18 \pm .02$ | $1.53 \pm .04$ |
| Comon, eq. (4) | $1.414 \pm .02$ | $.14 \pm .015$ | $.17 \pm .02$ | $3.0 \pm 2.5$ |
| Estimate (5) - 4th order | $1.414 \pm .02$ | $.13 \pm .015$ | $.17 \pm .02$ | $1.39 \pm .05$ |
| Estimate (5) - 6th order | $1.414 \pm .02$ | $.092 \pm .001$ | $.16 \pm .02$ | $1.3 \pm .5$ |
| theoretical value | $1.419$ | $.0$ | $.153$ | |

Table 1: Entropy values for different distributions sampled with $N = 1000$ data points and the different estimation methods explained in the text. The standard deviations are obtained by multiple repetition of the experiment.

1995). A general class of volume conserving transformations are the symplectic maps (Abraham & Marsden, 1978). An interesting and for our purpose important fact is that any symplectic transformation can be expressed in terms of a scalar function. And in turn any scalar function defines a symplectic map. In (Parra et al., 1995) a non-reflecting symplectic transformation has been presented. But its implicit definition results in the need of solving a nonlinear equation for each data point. This leads to time consuming computations which limit in practice the applications to low dimensional problems (n$\leq$ 10). In this work reflecting symplectic transformations with an explicit definition are used to define a "feed-forward" volume conserving maps. The input and output space is divided in two partitions $\mathbf{x} = (\mathbf{x}_1, \mathbf{x}_2)$ and $\mathbf{y} = (\mathbf{y}_1, \mathbf{y}_2)$, with $\mathbf{x}_1, \mathbf{x}_2, \mathbf{y}_1, \mathbf{y}_2 \in \Re^{n/2}$.

$$\mathbf{y}_1 = \mathbf{x}_1 - \frac{\partial P(\mathbf{x}_2)}{\partial \mathbf{x}_2} \quad , \quad \mathbf{y}_2 = \mathbf{x}_2 + \frac{\partial Q(\mathbf{y}_1)}{\partial \mathbf{y}_1} . \tag{6}$$

The structure of this symplectic map is represented in figure 1, right. Two scalar functions $P : \Re^{n/2} \mapsto \Re$ and $Q : \Re^{n/2} \mapsto \Re$ can be chosen arbitrarily. Note that for quadratic functions equation (6) represents a linear transformation. In order to have a general transformation we introduce for each of these scalar functions a 3-layer perceptron with nonlinear hidden units and a single linear output unit:

$$P(\mathbf{x}_2) = \mathbf{w}_2 \cdot g(W_2 \mathbf{x}_2) \quad , \quad Q(\mathbf{y}_1) = \mathbf{w}_1 \cdot g(W_1 \mathbf{y}_1) . \tag{7}$$

The scalar functions $P$ and $Q$ are parameterized by the network parameters $\mathbf{w}_1, \mathbf{w}_2 \in R^m$ and $W_1, W_2 \in R^m \times R^{n/2}$. The hidden-unit, nonlinear activation function $g$ applies to each component of the vectors $W_1 \mathbf{y}_1$ and $W_2 \mathbf{x}_2$ respectively. Because of the structure of equation (6) the output coordinates $\mathbf{y}_1$ depend only additively on the input coordinates $\mathbf{x}_1$. To obtain a more general nonlinear dependence a second symplectic layer has to be added.

To obtain factorial distributions the parameters of the map have to be trained. The approximations of the single coordinate entropies (4) or (5) are inserted in the mutual information optimization criterion (1). These approximations are expressed through moments in terms of the measured output data points. Therefore, the

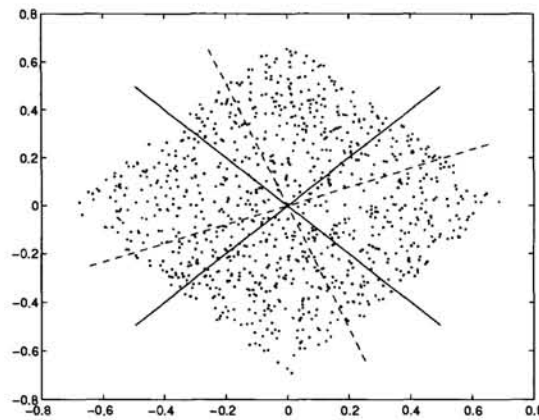

Figure 2: Sampled 2-dimensional squared uniform distribution rotated by $\pi/4$. Solid lines represent the directions found by any of the higher order techniques explained in the text. Dashed lines represent directions calculated by linear PCA. (This result is arbitrary and varies with noise).

gradient of these expressions with respect to parameters of the map can be computed in principle. For that matter different kinds of averages need to be computed. Even though, the computational complexity is not substantially increased compared with the efficient minimum variances criterion (2), the complexity of the algorithm increases considerably. Therefore, we applied an optimization algorithm that does not require any gradient information. The simple stochastic and parallel update algorithm ALOPEX (Unnikrishnan & Venugopal, 1994) was used.

## 4    Experiments

As explained above, finding the correct statistical independent directions of a rotated two dimensional uniform distribution causes problems for techniques which include only second order statistic. The statistical independent coordinates are simply the axes parallel to the edges of the distribution (see figure 2). A rotation *i.e.* a linear transformation suffices for this task. The covariance matrix of the data is diagonal for any rotation of the squared distribution and, hence, does not provide any information about the correct orientation of the square. It is well known, that PCA fails to find in the case of non-Gaussian distributions the statistical independent coordinates. Similarly the Gaussian upper bound technique (2) is not capable to minimize the mutual information in this case. Instead, with any one of the higher order criteria explained in the previous section one finds the appropriate coordinates for any linearly transformed multi-dimensional uniform distribution. This has been observed empirically for a series of setups. The symplectic map was restricted in this experiments to linearity by using square scalar functions.

The second example shows that the proposed technique in fact finds nonlinear relations between the input coordinates. An one-dimensional signal distributed according to the distribution of figure 1 was nonlinearly transformed into a two-

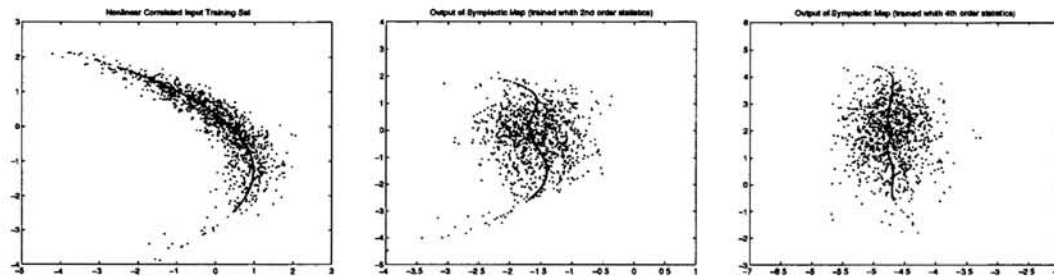

Figure 3: Symplectic map trained with 4th and 2nd order statistics corresponding to the equations (5) and (2) respectively. Left: input distribution. The line at the center of the distribution gives the nonlinear transformed noiseless signal distributed according to the distribution shown in figure 1. Center and Right: Output distribution of the symplectic map corresponding to the 4th order (right) and 2nd order (center) criterion.

dimensional signal and corrupted with additive noise, leading to the distribution shown in figure 3, left. The task of finding statistical independent coordinates has been tackled by an explicit symplectic transformation with $n = 2$ and $m = 6$. On figure 3 the different results for the optimization according to the Gaussian upper bound criterion (2) and the approximated entropy criterion (5) are shown. Obviously considering higher order statistics in fact improves the result by finding the better representation of the nonlinear dependency.

# Reference

Abraham, R., & Marsden, J. (1978). *Foundations of Mechanics* The Benjamin-Cummings Publishing Company, Inc., London.

Comon, P. (1994). Independent component analysis, A new concept *Signal Processing, 36*, 287–314.

Deco, G., & Brauer, W. (1994). Higher Order Statistical Decorrelation by Volume Concerning Nonlinear Maps. *Neural Networks, ?* submitted.

Deco, G., & Schürman, B. (1994). Learning Time Series Evolution by Unsupervised Extraction of Correlations. *Physical Review E, ?* submitted.

Kendall, M. G., & Stuart, A. (1969). *The Advanced Theory of Statistics* (3 edition)., Vol. 1. Charles Griffin and Company Limited, London.

Papoulis, A. (1991). *Probability, Random Variables, and Stochastic Processes.* Third Edition, McGraw-Hill, New York.

Parra, L., Deco, G., & Miesbach, S. (1995). Redundancy reduction with information-preserving nonlinear maps. *Network, 6*(1), 61–72.

Unnikrishnan, K., P., & Venugopal, K., P. (1994). Alopex: A Correlation-Based Learning Algorithm for Feedforward and Recurrent Neural Networks. *Neural Computation, 6*(3), 469–490.
